# Large Scale Bayes Point Machines

**Ralf Herbrich**
Statistics Research Group
Computer Science Department
Technical University of Berlin
*ralfh@cs.tu-berlin.de*

**Thore Graepel**
Statistics Research Group
Computer Science Department
Technical University of Berlin
*guru@cs.tu-berlin.de*

## Abstract

The concept of averaging over classifiers is fundamental to the Bayesian analysis of learning. Based on this viewpoint, it has recently been demonstrated for linear classifiers that the centre of mass of version space (the set of all classifiers consistent with the training set) — also known as the *Bayes point* — exhibits excellent generalisation abilities. However, the billiard algorithm as presented in [4] is restricted to small sample size because it requires $\mathcal{O}\left(m^2\right)$ of memory and $\mathcal{O}\left(N \cdot m^2\right)$ computational steps where $m$ is the number of training patterns and $N$ is the number of random draws from the posterior distribution. In this paper we present a method based on the simple perceptron learning algorithm which allows to overcome this algorithmic drawback. The method is algorithmically simple and is easily extended to the multi-class case. We present experimental results on the MNIST data set of handwritten digits which show that Bayes point machines (BPMs) are competitive with the current world champion, the support vector machine. In addition, the computational complexity of BPMs can be tuned by varying the number of samples from the posterior. Finally, rejecting test points on the basis of their (approximative) posterior probability leads to a rapid decrease in generalisation error, e.g. 0.1% generalisation error for a given rejection rate of 10%.

## 1 Introduction

Kernel machines have recently gained a lot of attention due to the popularisation of the support vector machine (SVM) [13] with a focus on classification and the revival of Gaussian Processes (GP) for regression [15]. Subsequently, SVMs have been modified to handle regression [12] and GPs have been adapted to the problem of classification [8]. Both schemes essentially work in the same function space that is characterised by kernels (SVM) and covariance functions (GP), respectively. While the formal similarity of the two methods is striking the underlying paradigms of inference are very different. The SVM was inspired by results from statistical/PAC learning theory while GPs are usually considered in a Bayesian framework. This ideological clash can be viewed as a continuation in machine learning of the by now classical disagreement between Bayesian and frequentistic statistics. With

regard to algorithmics the two schools of thought appear to favour two different methods of learning and predicting: the SVM community — as a consequence of the formulation of the SVM as a quadratic programming problem — focuses on learning as optimisation while the Bayesian community favours sampling schemes based on the Bayesian posterior. Of course there exists a strong relationship between the two ideas, in particular with the Bayesian maximum a posteriori (MAP) estimator being the solution of an optimisation problem. Interestingly, the two viewpoints have recently been reconciled theoretically in the so-called PAC-Bayesian framework [5] that combines the idea of a Bayesian prior with PAC-style performance guarantees and has been the basis of the so far tightest margin bound for SVMs [3]. In practice, optimisation based algorithms have the advantage of a unique, deterministic solution and the availability of the cost function as an indicator for the quality of the solution. In contrast, Bayesian algorithms based on sampling and voting are more flexible and have the so-called "anytime" property, providing a relatively good solution at any point in time. Often, however, they suffer from the computational costs of sampling the Bayesian posterior.

In this contribution we review the idea of the Bayes point machine (BPM) as an approximation to Bayesian inference for linear classifiers in kernel space in Section 2. In contrast to the GP viewpoint we do not define a Gaussian prior on the length $\|\mathbf{w}\|_{\mathcal{K}}$ of the weight vector. Instead, we only consider weight vectors of length $\|\mathbf{w}\|_{\mathcal{K}} = 1$ because it is only the spatial direction of the weight vector that matters for classification. It is then natural to define a uniform prior on the resulting ball-shaped hypothesis space. Hence, we determine the centre of mass ("Bayes point") of the resulting posterior that is uniform in version space, i.e. in the zero training error region. While the version space could be sampled using some form of Gibbs sampling (see, e.g. [6] for an overview) or an ergodic dynamic system such as a billiard [4] we suggest to use the perceptron algorithm trained on permutations of the training set for sampling in Section 3. This extremely simple sampling scheme proves to be efficient enough to make the BPM applicable to large data sets. We demonstrate this fact in Section 4 on the well-known MNIST data set containing 60 000 samples of handwritten digits and show how an approximation to the posterior probability of classification provided by the BPM can even be used for test-point rejection leading to a great reduction in generalisation error on the remaining samples.

We denote $n$–tuples by italic bold letters (e.g. $\boldsymbol{x} = (x_1, \ldots, x_n)$), vectors by roman bold letters (e.g. $\mathbf{x}$), random variables by sans serif font (e.g. $\mathsf{X}$) and vector spaces by calligraphic capitalised letters (e.g. $\mathcal{X}$). The symbols $\mathsf{P}, \mathsf{E}$ and $\mathsf{I}$ denote a probability measure, the expectation of a random variable and the indicator function, respectively.

## 2   Bayes Point Machines

Let us consider the task of classifying patterns $x \in \mathcal{X}$ into one of the two classes $y \in \mathcal{Y} = \{-1, +1\}$ using functions $h : \mathcal{X} \to \mathcal{Y}$ from a given set $\mathcal{H}$ known as the hypothesis space. In this paper we shall only be concerned with linear classifiers:

$$\mathcal{H} = \left\{ x \mapsto \text{sign} \left( \langle \phi(x), \mathbf{w} \rangle_{\mathcal{K}} \right) \mid \mathbf{w} \in \mathcal{W} \right\}, \quad \mathcal{W} = \left\{ \mathbf{w} \in \mathcal{K} \mid \|\mathbf{w}\|_{\mathcal{K}} = 1 \right\}, \quad (1)$$

where $\phi : \mathcal{X} \to \mathcal{K} \subseteq \ell_2^n$ is known[1] as the feature map and has to fixed beforehand. If all that is needed for learning and classification are the inner products $\langle \cdot, \cdot \rangle_{\mathcal{K}}$ in the feature space $\mathcal{K}$, it is convenient to specify $\phi$ only by its inner product function

$k : \mathcal{X} \times \mathcal{X} \to \mathbb{R}$ known as the *kernel*, i.e.

$$\forall x, x' \in \mathcal{X} : \qquad k\left(x, x'\right) = \left\langle \phi\left(x\right), \phi\left(x'\right)\right\rangle_{\mathcal{K}} .$$

For simplicity, let us assume that there exists a classifier[2] $\mathbf{w}^* \in \mathcal{W}$ that labels all our data, i.e.

$$\mathbf{P}_{\mathsf{Y}|\mathsf{X}=x,\mathsf{W}=\mathbf{w}^*}\left(y\right) = \mathbf{I}_{h_{\mathbf{w}^*}(x)=y} . \tag{2}$$

This assumption can easily be relaxed by introducing slack variables as done in the soft margin variant of the SVM. Then given a training set $z = (x, y)$ of $m$ points $x_i$ together with their classes $y_i$ assigned by $h_{\mathbf{w}^*}$ drawn iid from an unknown data distribution $\mathbf{P}_{\mathsf{Z}} = \mathbf{P}_{\mathsf{Y}|\mathsf{X}}\mathbf{P}_{\mathsf{X}}$ we can assume the existence of a *version space* $V(z)$, i.e. the set of all classifiers $\mathbf{w} \in \mathcal{W}$ consistent with $z$:

$$V\left(z\right) = \left\{\mathbf{w} \in \mathcal{W} \mid \forall\left(x_i, y_i\right) \in z : h_{\mathbf{w}}\left(x_i\right) = y_i\right\} . \tag{3}$$

In a Bayesian spirit we incorporate all of our prior knowledge about $\mathbf{w}^*$ into a *prior distribution* $\mathbf{P}_{\mathsf{W}}$ over $\mathcal{W}$. In the absence of any a priori knowledge we suggest a uniform prior over the spatial direction of weight vectors $\mathbf{w}$. Now, given the training set $z$ we update our prior belief by Bayes' formula, i.e.

$$
\begin{aligned}
\mathbf{P}_{\mathsf{W}|\mathsf{Z}^m=z}\left(\mathbf{w}\right) &= \frac{\mathbf{P}_{\mathsf{Z}^m|\mathsf{W}=\mathbf{w}}\left(z\right)\mathbf{P}_{\mathsf{W}}\left(\mathbf{w}\right)}{\mathbf{E}_{\mathsf{W}}\left[\mathbf{P}_{\mathsf{Z}^m|\mathsf{W}=\mathbf{w}}\left(z\right)\right]} = \frac{\prod_{i=1}^m \mathbf{P}_{\mathsf{Y}|\mathsf{X}=x_i,\mathsf{W}=\mathbf{w}}\left(y_i\right)\mathbf{P}_{\mathsf{W}}\left(\mathbf{w}\right)}{\mathbf{E}_{\mathsf{W}}\left[\prod_{i=1}^m \mathbf{P}_{\mathsf{Y}|\mathsf{X}=x_i,\mathsf{W}=\mathbf{w}}\left(y_i\right)\right]} \\
&= \begin{cases} \frac{\mathbf{P}_{\mathsf{W}}(\mathbf{w})}{\mathbf{P}_{\mathsf{W}}(V(z))} & \text{if } \mathbf{w} \in V\left(z\right) \\ 0 & \text{otherwise} \end{cases} ,
\end{aligned}
$$

where the first line follows from the independence and the fact that $x$ has no dependence on $\mathbf{w}$ and the second line follows from (2) and (3). The Bayesian classification of a novel test point $x$ is then given by

$$
\begin{aligned}
Bayes_z\left(x\right) &= \operatorname{argmax}_{y \in \mathcal{Y}} \mathbf{P}_{\mathsf{W}|\mathsf{Z}^m=z}\left(\left\{h_{\mathbf{w}}\left(x\right) = y\right\}\right) \\
&= \operatorname{sign}\left(\mathbf{E}_{\mathsf{W}|\mathsf{Z}^m=z}\left[h_{\mathbf{w}}\left(x\right)\right]\right) \\
&= \operatorname{sign}\left(\mathbf{E}_{\mathsf{W}|\mathsf{Z}^m=z}\left[\operatorname{sign}\left(\langle\mathbf{x},\mathbf{W}\rangle_{\mathcal{K}}\right)\right]\right) .
\end{aligned}
$$

Unfortunately, the strategy $Bayes_z$ is in general *not* contained in the set $\mathcal{H}$ of classifiers considered beforehand. Since $\mathbf{P}_{\mathsf{W}|\mathsf{Z}^m=z}$ is only non-zero inside version space, it has been suggested to use the centre of mass $\mathbf{w}_{\mathrm{cm}}$ as an approximation for $Bayes_z$, i.e.

$$
\begin{aligned}
h_{\mathrm{bp}}\left(x\right) &= \operatorname{sign}\left(\mathbf{E}_{\mathsf{W}|\mathsf{Z}^m=z}\left[\langle\mathbf{x},\mathbf{W}\rangle_{\mathcal{K}}\right]\right) \\
&= \operatorname{sign}\left(\langle\mathbf{x},\mathbf{w}_{\mathrm{cm}}\rangle_{\mathcal{K}}\right) , \\
\mathbf{w}_{\mathrm{cm}} &= \mathbf{E}_{\mathsf{W}|\mathsf{Z}^m=z}\left[\mathbf{W}\right] . \tag{4}
\end{aligned}
$$

This classifier is called the *Bayes point*. In a previous work [4] we calculated $\mathbf{w}_{\mathrm{cm}}$ using a first order Markov chain based on a billiard-like algorithm (see also [10]). We entered the version space $V(z)$ using a perceptron algorithm and started playing billiards in version space $V(z)$ thus creating a sequence of pseudo-random samples $\mathbf{w}_i$ due to the chaotic nature of the billiard dynamics. Playing billiards in $V(z)$ is possible because each training point $(x_i, y_i) \in z$ defines a hyperplane $\left\{\mathbf{w} \in \mathcal{W} \mid y_i \langle\mathbf{x}_i,\mathbf{w}\rangle_{\mathcal{K}} = 0\right\} \subseteq \mathcal{W}$. Hence, the version space is a convex polyhedron on the surface of $\mathcal{W}$. After $N$ bounces of the billiard ball the Bayes point was estimated by

$$\widehat{\mathbf{w}}_{\mathrm{cm}} = \frac{1}{N}\sum_{i=1}^N \mathbf{w}_i .$$

Although this algorithm shows excellent generalisation performance when compared to state-of-the art learning algorithms like support vector machines (SVM) [13], its effort scales like $\mathcal{O}\left(m^2\right)$ and $\mathcal{O}\left(N \cdot m^2\right)$ in terms of memory and computational requirements, respectively.

## 3  Sampling the Version Space

Clearly, all we need for estimating the Bayes point (4) is a set of classifiers $\mathbf{w}$ drawn uniformly from $V\left(\mathbf{z}\right)$. In order to save computational resources it might be advantageous to achieve a uniform sample only approximately. The classical perceptron learning algorithm offers the possibility to obtain up to $m!$ different classifiers in version space simply by learning on different permutations of the training set. Given a permutation $\Pi : \{1, \ldots, m\} \rightarrow \{1, \ldots, m\}$ the perceptron algorithm works as follows:

1. Start with $\mathbf{w}_0 = \mathbf{0}$ and $t = 0$.
2. For all $i \in \{1, \ldots, m\}$, if $y_{\Pi(i)} \left\langle \mathbf{x}_{\Pi(i)}, \mathbf{w}_t \right\rangle_{\mathcal{K}} \leq 0$ then $\mathbf{w}_{t+1} = \mathbf{w}_t + y_{\Pi(i)} \mathbf{x}_{\Pi(i)}$ and $t \leftarrow t + 1$.
3. Stop, if for all $i \in \{1, \ldots, m\}$, $y_{\Pi(i)} \left\langle \mathbf{x}_{\Pi(i)}, \mathbf{w}_t \right\rangle_{\mathcal{K}} > 0$.

A classical theorem due to Novikoff [7] guarantees the convergence of this procedure and furthermore provides an upper bound on the number $t$ of mistakes needed until convergence. More precisely, if there exists a classifier $\mathbf{w}_{\text{SVM}}$ with margin

$$\gamma_{\mathbf{z}}\left(\mathbf{w}_{\text{SVM}}\right) = \min_{(x_i, y_i) \in \mathbf{z}} \frac{y_i \left\langle \mathbf{x}_i, \mathbf{w}_{\text{SVM}} \right\rangle_{\mathcal{K}}}{\|\mathbf{w}_{\text{SVM}}\|_{\mathcal{K}}},$$

then the number of mistakes until convergence — which is an upper bound on the sparsity of the solution — is not more than $R^2\left(\boldsymbol{x}\right) \gamma_{\mathbf{z}}^{-2}\left(\mathbf{w}_{\text{SVM}}\right)$, where $R\left(\boldsymbol{x}\right)$ is the smallest real number such that $\forall x \in \boldsymbol{x} : \|\phi\left(x\right)\|_{\mathcal{K}} \leq R\left(\boldsymbol{x}\right)$. The quantity $\gamma_{\mathbf{z}}\left(\mathbf{w}_{\text{SVM}}\right)$ is maximised for the solution $\mathbf{w}_{\text{SVM}}$ found by the SVM, and whenever the SVM is theoretically justified by results from learning theory (see [11, 13]) the ratio $d = R^2\left(\boldsymbol{x}\right) \gamma_{\mathbf{z}}^{-2}\left(\mathbf{w}_{\text{SVM}}\right)$ is considerably less than $m$, say $d \ll m$.

Algorithmically, we can benefit from this sparsity by the following "trick": since

$$\mathbf{w} = \sum_{i=1}^{m} \alpha_i \mathbf{x}_i$$

all we need to store is the $m$–dimensional vector $\boldsymbol{\alpha}$. Furthermore, we keep track of the $m$–dimensional vector $\mathbf{o}$ of real valued outputs

$$o_i = y_i \left\langle \mathbf{x}_i, \mathbf{w}_t \right\rangle_{\mathcal{K}} = \sum_{j=1}^{m} \alpha_j k\left(x_i, x_j\right)$$

of the current solution at the $i$–th training point. By definition, in the beginning $\boldsymbol{\alpha} = \mathbf{o} = \mathbf{0}$. Now, if $o_i \leq 0$ we update $\alpha_i$ by $\alpha_i + y_i$ and update $\mathbf{o}$ by $o_j \leftarrow o_j + y_i k\left(x_i, x_j\right)$ which requires only $m$ kernel calculations. In summary, the memory requirement of this algorithm is $2m$ and the number of kernel calculations is not more than $d \cdot m$. As a consequence, the computational requirement of this algorithm is no more than the computational requirement for the evaluation of the margin $\gamma_{\mathbf{z}}\left(\mathbf{w}_{\text{SVM}}\right)$! We suggest to use this efficient perceptron learning algorithm in order to obtain samples $\mathbf{w}_i$ for the computation of the Bayes point by (4).

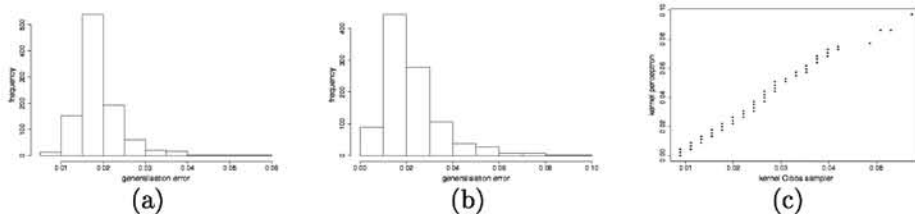

|  (a)  |  (b)  |  (c)  |

Figure 1: **(a)** Histogram of generalisation errors (estimated on a test set) using a kernel Gibbs sampler. **(b)** Histogram of generalisation errors (estimated on a test set) using a kernel perceptron. **(c)** QQ plot of distributions (a) and (b). The straight line indicates that both distribution are very similar.

In order to investigate the usefulness of this approach experimentally, we compared the distribution of generalisation errors of samples obtained by perceptron learning on permuted training sets (as suggested earlier by [14]) with samples obtained by a full Gibbs sampling [2]. For computational reasons, we used only 188 training patterns and 453 test patterns of the classes "1" and "2" from the MNIST data set[3]. In Figure 1 (a) and (b) we plotted the distribution over 1000 random samples using the kernel[4]

$$k\left(x, x'\right) = \left(\langle x, x'\rangle_{\mathcal{X}} + 1\right)^5 . \qquad (5)$$

Using a quantile-quantile (QQ) plot technique we can compare both distributions in one graph (see Figure 1 (c)). These plots suggest that by simple permutation of the training set we are able to obtain a sample of classifiers exhibiting the same generalisation error distribution as with time-consuming Gibbs sampling.

## 4  Experimental Results

In our large scale experiment we used the full MNIST data set with 60 000 training examples and 10 000 test examples of $28 \times 28$ grey value images of handwritten digits. As input vector $x$ we used the 784 dimensional vector of grey values. The images were labelled by one of the ten classes "0" to "1". For each of the ten classes $y = \{0, \ldots, 9\}$ we ran the perceptron algorithm $N = 10$ times each time labelling all training points of class $y$ by $+1$ and the remaining training points by $-1$. On an Ultra Sparc 10 each learning trial took approximately $20 - 30$ minutes. For the classification of a test image $x$ we calculated the real-valued output of all 100 different classifiers[5] by

$$f_i(x) = \frac{\langle \mathbf{x}, \mathbf{w}_i\rangle_{\mathcal{K}}}{\|\mathbf{w}_i\|_{\mathcal{K}} \|\mathbf{x}\|_{\mathcal{K}}} = \frac{\sum\limits_{j=1}^{m} (\boldsymbol{\alpha}_i)_j\, k\,(x_j, x)}{\sqrt{\sum\limits_{j=1}^{m} \sum\limits_{l=1}^{m} (\boldsymbol{\alpha}_i)_j\, (\boldsymbol{\alpha}_i)_l\, k\,(x_j, x_l)} \sqrt{k\,(x, x)}},$$

where we used the kernel $k$ given by (5). $(\boldsymbol{\alpha}_i)_j$ refers to the expansion coefficient corresponding to the $i$–th classifier and the $j$–th data point. Now, for each of the

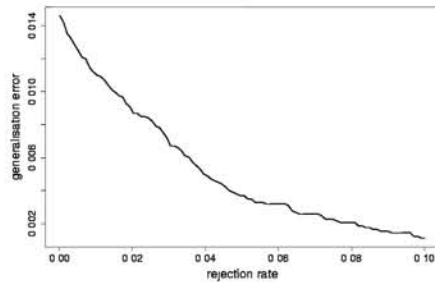

| rejection rate | generalisation error |
|:---:|:---:|
| 0% | 1.46% |
| 1% | 1.10% |
| 2% | 0.87% |
| 3% | 0.67% |
| 4% | 0.49% |
| 5% | 0.37% |
| 6% | 0.32% |
| 7% | 0.26% |
| 8% | 0.21% |
| 9% | 0.14% |
| 10% | 0.11% |

Figure 2: Generalisation error as a function of the rejection rate for the MNIST data set. The SVM achieved 1.4% without rejection as compared to 1.46% for the BPM. Note that by rejection based on the real-valued output the generalisation error could be reduced to 0.1% indicating that this measure is related to the probability of misclassification of single test points.

ten classes we calculated the real-valued decision of the Bayes point $\mathbf{w}_y$ by

$$f_{\mathrm{bp},y}\left(x\right) = \frac{1}{N}\sum_{i=1}^{N} f_{i+yN}\left(x\right) .$$

In a Bayesian spirit, the final decision was carried out by

$$h_{\mathrm{bp}}\left(x\right) = \mathrm{argmax}_{y\in\{0,\ldots,9\}} f_{\mathrm{bp},y}\left(x\right) .$$

Note that $f_{\mathrm{bp},y}\left(x\right)$ [9] can be interpreted as an (unnormalised) approximation of the posterior probability that $x$ is of class $y$ when restricted to the function class (1). In order to test the dependence of the generalisation error on the magnitude $\max_y f_{\mathrm{bp},y}\left(x\right)$ we fixed a certain rejection rate $r \in [0,1]$ and rejected the set of $r \cdot 10\,000$ test points with the smallest value of $\max_y f_{\mathrm{bp},y}\left(x\right)$. The resulting plot is depicted in Figure 2.

As can be seen from this plot, even without rejection the Bayes point has excellent generalisation performance[6]. Furthermore, rejection based on the real-valued output $f_{\mathrm{bp}}\left(x\right)$ turns out to be excellent thus reducing the generalisation error to 0.1%. One should also bear in mind that the learning time for this simple algorithm was comparable to that of SVMs.

A very advantageous feature of our approach as compared to SVMs are its adjustable time and memory requirements and the "anytime" availability of a solution due to sampling. If the training set grows further and we are not able to spend more time with learning, we can adjust the number $N$ of samples used at the price of slightly worse generalisation error.

## 5  Conclusion

In this paper we have presented an algorithm for approximating the Bayes point by rerunning the classical perceptron algorithm with a permuted training set. Here we

particularly exploited the sparseness of the solution which *must* exist whenever the success of the SVM is theoretically justified. The restriction to the zero training error case can be overcome by modifying the kernel as

$$k_\lambda\left(x, x'\right) = k\left(x, x'\right) + \lambda \cdot \mathbf{I}_{x=x'}.$$

This technique is well known and was already suggested by Vapnik in 1995 (see [1]). Another interesting question raised by our experimental findings is the following: By how much is the distribution of generalisation errors over random samples from version space related to the distribution of generalisation errors of the up to $m!$ different classifiers found by the classical perceptron algorithm?

**Acknowledgements** We would like to thank Bob Williamson for helpful discussions and suggestions on earlier drafts. Parts of this work were done during a research stay of both authors at the ANU Canberra.

## Footnotes

[1] For notational convenience we shall abbreviate $\phi(x)$ by $\mathbf{x}$. This should not be confused with the set $\boldsymbol{x}$ of training points.

[2]We synonymously call $h \in \mathcal{H}$ and $\mathbf{w} \in \mathcal{W}$ a *classifier* because there is a one-to-one correspondence between the two by virtue of (1).

[3]available at http://www.research.att.com/~yann/ocr/mnist/.

[4]We decided to use this kernel because it showed excellent generalisation performance when using the support vector machine.

[5]For notational simplicity we assume that the first $N$ classifiers are classifiers for the class "0", the next $N$ for class "1" and so on.

[6]Note that the best know result on this data set if 1.1 achieved with a polynomial kernel of degree four. Nonetheless, for reason of fairness we compared the results of both algorithms using the *same* kernel.

# References

[1] C. Cortes and V. Vapnik. Support Vector Networks. *Machine Learning*, 20:273–297, 1995.

[2] T. Graepel and R. Herbrich. The kernel Gibbs sampler. In *Advances in Neural Information System Processing 13*, 2001.

[3] R. Herbrich and T. Graepel. A PAC-Bayesian margin bound for linear classifiers: Why SVMs work. In *Advances in Neural Information System Processing 13*, 2001.

[4] R. Herbrich, T. Graepel, and C. Campbell. Robust Bayes Point Machines. In *Proceedings of ESANN 2000*, pages 49–54, 2000.

[5] D. A. McAllester. Some PAC Bayesian theorems. In *Proceedings of the Eleventh Annual Conference on Computational Learning Theory*, pages 230–234, Madison, Wisconsin, 1998.

[6] R. M. Neal. Markov chain monte carlo method based on 'slicing' the density function. Technical report, Department of Statistics, University of Toronto, 1997. TR–9722.

[7] A. Novikoff. On convergence proofs for perceptrons. In *Report at the Symposium on Mathematical Theory of Automata*, pages 24–26, Politechnical Institute Brooklyn, 1962.

[8] M. Opper and O. Winther. Gaussian processes for classification: Mean field algorithms. *Neural Computation*, 12(11), 2000.

[9] J. Platt. Probabilities for SV machines. In *Advances in Large Margin Classifiers*, pages 61–74. MIT Press, 2000.

[10] P. Ruján and M. Marchand. Computing the bayes kernel classifier. In *Advances in Large Margin Classifiers*, pages 329–348. MIT Press, 2000.

[11] J. Shawe-Taylor, P. L. Bartlett, R. C. Williamson, and M. Anthony. Structural risk minimization over data–dependent hierarchies. *IEEE Transactions on Information Theory*, 44(5):1926–1940, 1998.

[12] A. J. Smola. *Learning with Kernels*. PhD thesis, Technische Universität Berlin, 1998.

[13] V. Vapnik. *The Nature of Statistical Learning Theory*. Springer, 1995.

[14] T. Watkin. Optimal learning with a neural network. *Europhysics Letters*, 21:871–877, 1993.

[15] C. Williams. Prediction with Gaussian Processes: From linear regression to linear prediction and beyond. Technical report, Neural Computing Research Group, Aston University, 1997. NCRG/97/012.